# Stochastic Gradient Descent with Only One Projection

**Mehrdad Mahdavi**[†]**, Tianbao Yang**[‡]**, Rong Jin**[†]**, Shenghuo Zhu**[⋆]**, and Jinfeng Yi**[†]

[†]Dept. of Computer Science and Engineering, Michigan State University, MI, USA
[‡]Machine Learning Lab, GE Global Research, CA, USA
[⋆]NEC Laboratories America, CA, USA
[†]{mahdavim,rongjin,yijinfen}@msu.edu,[‡]tyang@ge.com,[⋆]zsh@nec-labs.com

## Abstract

Although many variants of stochastic gradient descent have been proposed for large-scale convex optimization, most of them require projecting the solution at *each* iteration to ensure that the obtained solution stays within the feasible domain. For complex domains (e.g., positive semidefinite cone), the projection step can be computationally expensive, making stochastic gradient descent unattractive for large-scale optimization problems. We address this limitation by developing novel stochastic optimization algorithms that do not need intermediate projections. Instead, only one projection at the last iteration is needed to obtain a feasible solution in the given domain. Our theoretical analysis shows that with a high probability, the proposed algorithms achieve an $O(1/\sqrt{T})$ convergence rate for general convex optimization, and an $O(\ln T/T)$ rate for strongly convex optimization under mild conditions about the domain and the objective function.

## 1 Introduction

With the increasing amount of data that is available for training, it becomes an urgent task to devise efficient algorithms for optimization/learning problems with unprecedented sizes. Online learning algorithms, such as celebrated Stochastic Gradient Descent (SGD) [16, 2] and its online counterpart Online Gradient Descent (OGD) [22], despite of their slow rate of convergence compared with the batch methods, have shown to be very effective for large scale and online learning problems, both theoretically [16, 13] and empirically [19]. Although a large number of iterations is usually needed to obtain a solution of desirable accuracy, the lightweight computation per iteration makes SGD attractive for many large-scale learning problems.

To find a solution within the domain $\mathcal{K}$ that optimizes the given objective function $f(\mathbf{x})$, SGD computes an unbiased estimate of the gradient of $f(\mathbf{x})$, and updates the solution by moving it in the opposite direction of the estimated gradient. To ensure that the solution stays within the domain $\mathcal{K}$, SGD has to project the updated solution back into the $\mathcal{K}$ at *every iteration*. Although efficient algorithms have been developed for projecting solutions into special domains (e.g., simplex and $\ell_1$ ball [6, 14]); for complex domains, such as a positive semidefinite (PSD) cone in metric learning and bounded trace norm matrices in matrix completion (more examples of complex domains can be found in [10] and [11]), the projection step requires solving an expensive convex optimization, leading to a high computational cost per iteration and consequently making SGD unappealing for large-scale optimization problems over such domains. For instance, projecting a matrix into a PSD cone requires computing the full eigen-decomposition of the matrix, whose complexity is cubic in the size of the matrix.

The central theme of this paper is to develop a SGD based method that does not require projection at each iteration. This problem was first addressed in a very recent work [10], where the authors extended Frank-Wolfe algorithm [7] for online learning. But, one main shortcoming of the algo-

rithm proposed in [10] is that it has a slower convergence rate (i.e., $O(T^{-1/3})$) than a standard SGD algorithm (i.e., $O(T^{-1/2})$). In this work, we demonstrate that a properly modified SGD algorithm can achieve the optimal convergence rate of $O(T^{-1/2})$ using only **ONE** projection for general stochastic convex optimization problem. We further develop an SGD based algorithm for strongly convex optimization that achieves a convergence rate of $O(\ln T/T)$, which is only a logarithmic factor worse than the optimal rate [9]. The key idea of both algorithms is to appropriately penalize the intermediate solutions when they are outside the domain. With an appropriate design of penalization mechanism, the average solution $\widehat{\mathbf{x}}_T$ obtained by the SGD after $T$ iterations will be very close to the domain $\mathcal{K}$, even without intermediate projections. As a result, the final feasible solution $\widetilde{\mathbf{x}}_T$ can be obtained by projecting $\widehat{\mathbf{x}}_T$ into the domain $\mathcal{K}$, the only projection that is needed for the entire algorithm. We note that our approach is very different from the previous efforts in developing projection free convex optimization algorithms (see [8, 12, 11] and references therein), where the key idea is to develop appropriate *updating* procedures to restore the feasibility of solutions at every iteration.

We close this section with a statement of contributions and main results made by the present work:

- We propose a stochastic gradient descent algorithm for general convex optimization that introduces a Lagrangian multiplier to penalize the solutions outside the domain and performs primal-dual updating. The proposed algorithm achieves the optimal convergence rate of $O(1/\sqrt{T})$ with only one projection;
- We propose a stochastic gradient descent algorithm for strongly convex optimization that constructs the penalty function using a smoothing technique. This algorithm attains an $O(\ln T/T)$ convergence rate with only one projection.

## 2 Related Works

Generally, the computational complexity of the projection step in SGD has seldom been taken into account in the literature. Here, we briefly review the previous works on projection free convex optimization, which is closely related to the theme of this study. For some specific domains, efficient algorithms have been developed to circumvent the high computational cost caused by projection step at each iteration of gradient descent methods. The main idea is to select an appropriate direction to take from the current solution such that the next solution is guaranteed to stay within the domain. Clarkson [5] proposed a sparse greedy approximation algorithm for convex optimization over a simplex domain, which is a generalization of an old algorithm by Frank and Wolfe [7] (a.k.a conditional gradient descent [3]). Zhang [21] introduced a similar sequential greedy approximation algorithm for certain convex optimization problems over a domain given by a convex hull. Hazan [8] devised an algorithm for approximately maximizing a concave function over a trace norm bounded PSD cone, which only needs to compute the maximum eigenvalue and the corresponding eigenvector of a symmetric matrix. Ying et al. [20] formulated the distance metric learning problems into eigenvalue maximization and proposed an algorithm similar to [8].

Recently, Jaggi [11] put these ideas into a general framework for convex optimization with a general convex domain. Instead of projecting the intermediate solution into a complex convex domain, Jaggi's algorithm solves a linearized problem over the same domain. He showed that Clark's algorithm , Zhang's algorithm and Hazan's algorithm discussed above are special cases of his general algorithm for special domains. It is important to note that all these algorithms are designed for batch optimization, not for stochastic optimization, which is the focus of this work.

Our work is closely related to the online Frank-Wolfe (OFW) algorithm proposed in [10]. It is a projection free online learning algorithm, built on the the assumption that it is possible to efficiently minimize a linear function over the complex domain. One main shortcoming of the OFW algorithm is that its convergence rate for general stochastic optimization is $O(T^{-1/3})$, significantly slower than that of a standard stochastic gradient descent algorithm (i.e., $O(T^{-1/2})$). It achieves a convergence rate of $O(T^{-1/2})$ only when the objective function is smooth, which unfortunately does not hold for many machine learning problems where either a non-smooth regularizer or a non-smooth loss function is used. Another limitation of OFW is that it assumes a linear optimization problem over the domain $\mathcal{K}$ can be solved efficiently. Although this assumption holds for some specific domains as discussed in [10], but in many settings of practical interest, this may not be true. The proposed algorithms address the two limitations explicitly. In particular, we show that how two seemingly different modifications of the SGD can be used to avoid performing expensive projections with similar convergency rates as the original SGD method.

## 3 Preliminaries

Throughout this paper, we consider the following convex optimization problem:

$$\min_{\mathbf{x} \in \mathcal{K}} f(\mathbf{x}), \tag{1}$$

where $\mathcal{K}$ is a bounded convex domain. We assume that $\mathcal{K}$ can be characterized by an inequality constraint and without loss of generality is bounded by the unit ball, i.e.,

$$\mathcal{K} = \{\mathbf{x} \in \mathbb{R}^d : g(\mathbf{x}) \leq 0\} \subseteq \mathcal{B} = \{\mathbf{x} \in \mathbb{R}^d : \|\mathbf{x}\|_2 \leq 1\}, \tag{2}$$

where $g(\mathbf{x})$ is a convex constraint function. We assume that $\mathcal{K}$ has a non-empty interior, i.e., there exists $\mathbf{x}$ such that $g(\mathbf{x}) < 0$ and the optimal solution $\mathbf{x}^*$ to (1) is in the interior of the unit ball $\mathcal{B}$, i.e., $\|\mathbf{x}^*\|_2 < 1$. Note that when a domain is characterized by multiple convex constraint functions, say $g_i(\mathbf{x}) \leq 0, i = 1, \ldots, m$, we can summarize them into one constraint $g(\mathbf{x}) \leq 0$, by defining $g(\mathbf{x})$ as $g(\mathbf{x}) = \max_{1 \leq i \leq m} g_i(\mathbf{x})$.

To solve the optimization problem in (1), we assume that the only information available to the algorithm is through a stochastic oracle that provides unbiased estimates of the gradient of $f(\mathbf{x})$. More precisely, let $\xi_1, \ldots, \xi_T$ be a sequence of independently and identically distributed (i.i.d) random variables sampled from an unknown distribution $P$. At each iteration $t$, given solution $\mathbf{x}_t$, the oracle returns $\widetilde{\nabla} f(\mathbf{x}_t; \xi_t)$, an unbiased estimate of the true gradient $\nabla f(\mathbf{x}_t)$, i.e., $\mathrm{E}_{\xi_t}[\widetilde{\nabla} f(\mathbf{x}_t, \xi_t)] = \nabla f(\mathbf{x}_t)$. The goal of the learner is to find an approximate optimal solution by making $T$ calls to this oracle.

Before proceeding, we recall a few definitions from convex analysis [17].

**Definition 1.** *A function $f(\mathbf{x})$ is a G-Lipschitz continuous function w.r.t a norm $\| \cdot \|$, if*

$$|f(\mathbf{x}_1) - f(\mathbf{x}_2)| \leq G\|\mathbf{x}_1 - \mathbf{x}_2\|, \forall \mathbf{x}_1, \mathbf{x}_2 \in \mathcal{B}. \tag{3}$$

In particular, a convex function $f(\mathbf{x})$ with a bounded (sub)gradient $\|\partial f(\mathbf{x})\|_* \leq G$ is $G$-Lipschitz continuous, where $\| \cdot \|_*$ is the dual norm to $\| \cdot \|$.

**Definition 2.** *A convex function $f(\mathbf{x})$ is $\beta$-strongly convex w.r.t a norm $\| \cdot \|$ if there exists a constant $\beta > 0$ (often called the modulus of strong convexity) such that, for any $\alpha \in [0, 1]$, it holds:*

$$f(\alpha\mathbf{x}_1 + (1 - \alpha)\mathbf{x}_2) \leq \alpha f(\mathbf{x}_1) + (1 - \alpha)f(\mathbf{x}_2) - \frac{1}{2}\alpha(1 - \alpha)\beta\|\mathbf{x}_1 - \mathbf{x}_2\|^2, \forall \mathbf{x}_1, \mathbf{x}_2 \in \mathcal{B}.$$

When $f(\mathbf{x})$ is differentiable, the strong convexity is equivalent to $f(\mathbf{x}_1) \geq f(\mathbf{x}_2) + \langle \nabla f(\mathbf{x}_2), \mathbf{x}_1 - \mathbf{x}_2 \rangle + \frac{\beta}{2}\|\mathbf{x}_1 - \mathbf{x}_2\|^2, \forall \mathbf{x}_1, \mathbf{x}_2 \in \mathcal{B}$. In the sequel, we use the standard Euclidean norm to define Lipschitz and strongly convex functions. Stochastic gradient descent method is an iterative algorithm and produces a sequence of solutions $\mathbf{x}_t, t = 1, \ldots, T$, by

$$\mathbf{x}_{t+1} = \Pi_{\mathcal{K}}(\mathbf{x}_t - \eta_t \widetilde{\nabla} f(\mathbf{x}_t, \xi_t)), \tag{4}$$

where $\{\eta_t\}_{t=1}^T$ is a sequence of step sizes, $\Pi_{\mathcal{K}}(\mathbf{x})$ is a projection operator that projects $\mathbf{x}$ into the domain $\mathcal{K}$, and $\widetilde{\nabla} f(\mathbf{x}, \xi_t)$ is an unbiased stochastic gradient of $f(\mathbf{x})$, for which we further assume bounded gradient variance as

$$\mathrm{E}_{\xi_t}[\exp(\|\widetilde{\nabla} f(\mathbf{x}, \xi_t) - \nabla f(\mathbf{x})\|_2^2/\sigma^2)] \leq \exp(1). \tag{5}$$

For general convex optimization, stochastic gradient descent methods can obtain an $O(1/\sqrt{T})$ convergence rate in expectation or in a high probability provided (5) [16]. As we mentioned in the Introduction section, SGD methods are computationally efficient only when the projection $\Pi_{\mathcal{K}}(\mathbf{x})$ can be carried out efficiently. The objective of this work is to develop computationally efficient stochastic optimization algorithms that are able to yield the same performance guarantee as the standard SGD algorithm but with only ONE projection when applied to the problem in (1).

## 4 Algorithms and Main Results

We now turn to extending the SGD method to the setting where only one projection is allowed to perform for the entire sequence of updating. The main idea is to incorporate the constraint function $g(\mathbf{x})$ into the objective function to penalize the intermediate solutions that are outside the domain. The result of the penalization is that, although the average solution obtained by SGD may not be feasible, it should be very close to the boundary of the domain. A projection is performed at the end of the iterations to restore the feasibility of the average solution.

---

**Algorithm 1** (SGDP-PD): SGD with ONE Projection by Primal Dual Updating

---

1: **Input**: a sequence of step sizes $\{\eta_t\}$, and a parameter $\gamma > 0$
2: **Initialization**: $\mathbf{x}_1 = \mathbf{0}$ and $\lambda_1 = 0$
3: **for** $t = 1, 2, \ldots, T$ **do**
4:      Compute $\mathbf{x}'_{t+1} = \mathbf{x}_t - \eta_t(\widetilde{\nabla} f(\mathbf{x}_t, \xi_t) + \lambda_t \nabla g(\mathbf{x}_t))$
5:      Update $\mathbf{x}_{t+1} = \mathbf{x}'_{t+1} / \max(\|\mathbf{x}'_{t+1}\|_2, 1)$,
6:      Update $\lambda_{t+1} = [(1 - \gamma \eta_t)\lambda_t + \eta_t g(\mathbf{x}_t)]_+$
7: **end for**
8: **Output:** $\widetilde{\mathbf{x}}_T = \Pi_{\mathcal{K}}(\widehat{\mathbf{x}}_T)$, where $\widehat{\mathbf{x}}_T = \sum_{t=1}^{T} \mathbf{x}_t / T$.

---

The key ingredient of proposed algorithms is to replace the projection step with the gradient computation of the constraint function defining the domain $\mathcal{K}$, which is significantly cheaper than projection step. As an example, when a solution is restricted to a PSD cone, i.e., $X \succeq 0$ where $X$ is a symmetric matrix, the corresponding inequality constraint is $g(X) = \lambda_{\max}(-X) \leq 0$, where $\lambda_{\max}(X)$ computes the largest eigenvalue of $X$ and is a convex function. In this case, $\nabla g(X)$ only requires computing the minimum eigenvector of a matrix, which is cheaper than a full eigenspectrum computation required at each iteration of the standard SGD algorithm to restore feasibility.

Below, we state a few assumptions about $f(\mathbf{x})$ and $g(\mathbf{x})$ often made in stochastic optimization as:

$$\textbf{A1} \qquad \|\nabla f(\mathbf{x})\|_2 \leq G_1, \quad \|\nabla g(\mathbf{x})\|_2 \leq G_2, \quad |g(\mathbf{x})| \leq C_2, \quad \forall \mathbf{x} \in \mathcal{B}, \qquad (6)$$

$$\textbf{A2} \qquad \mathrm{E}_{\xi_t}[\exp(\|\widetilde{\nabla} f(\mathbf{x}, \xi_t) - \nabla f(\mathbf{x})\|_2^2 / \sigma^2)] \leq \exp(1), \quad \forall \mathbf{x} \in \mathcal{B}. \qquad (7)$$

We also make the following mild assumption about the boundary of the convex domain $\mathcal{K}$ as:

$$\textbf{A3} \qquad \text{there exists a constant } \rho > 0 \text{ such that } \min_{g(\mathbf{x})=0} \|\nabla g(\mathbf{x})\|_2 \geq \rho. \qquad (8)$$

**Remark 1.** *The purpose of introducing assumption* **A3** *is to ensure that the optimal dual variable for the constrained optimization problem in (1) is well bounded from the above, a key factor for our analysis. To see this, we write the problem in (1) into a convex-concave optimization problem:*

$$\min_{\mathbf{x} \in \mathcal{B}} \max_{\lambda \geq 0} f(\mathbf{x}) + \lambda g(\mathbf{x}).$$

*Let $(\mathbf{x}_*, \lambda_*)$ be the optimal solution to the above convex-concave optimization problem. Since we assume $g(\mathbf{x})$ is strictly feasible, $\mathbf{x}_*$ is also an optimal solution to (1) due to the strong duality theorem [4]. Using the first order optimality condition, we have $\nabla f(\mathbf{x}_*) = -\lambda_* \nabla g(\mathbf{x}_*)$. Hence, $\lambda_* = 0$ when $g(\mathbf{x}_*) < 0$, and $\lambda_* = \|\nabla f(\mathbf{x}_*)\|_2 / \|\nabla g(\mathbf{x}_*)\|_2$ when $g(\mathbf{x}_*) = 0$. Under assumption* **A3**, *we have $\lambda_* \in [0, G_1/\rho]$.*

We note that, from a practical point of view, it is straightforward to verify that for many domains including PSD cone and Polytope, the gradient of the constraint function is lower bounded on the boundary and therefore assumption **A3** does not limit the applicability of the proposed algorithms for stochastic optimization. For the example of $g(X) = \lambda_{\max}(-X)$, the assumption **A3** implies $\min_{g(X)=0} \|\nabla g(X)\|_F = \|\mathbf{u}\mathbf{u}^\top\|_F = 1$, where $\mathbf{u}$ is an orthonomal vector representing the corresponding eigenvector of the matrix $X$ whose minimum eigenvalue is zero.

We propose two different ways of incorporating the constraint function into the objective function, which result in two algorithms, one for general convex and the other for strongly convex functions.

### 4.1 SGD with One Projection for General Convex Optimization

To incorporate the constraint function $g(\mathbf{x})$, we introduce a regularized Lagrangian function,

$$L(\mathbf{x}, \lambda) = f(\mathbf{x}) + \lambda g(\mathbf{x}) - \frac{\gamma}{2}\lambda^2, \quad \lambda \geq 0.$$

The summation of the first two terms in $L(\mathbf{x}, \lambda)$ corresponds to the Lagrangian function in dual analysis and $\lambda$ corresponds to a Lagrangian multiplier. A regularization term $-(\gamma/2)\lambda^2$ is introduced in $L(\mathbf{x}, \lambda)$ to prevent $\lambda$ from being too large. Instead of solving the constrained optimization problem in (1), we try to solve the following convex-concave optimization problem

$$\min_{\mathbf{x} \in \mathcal{B}} \max_{\lambda \geq 0} L(\mathbf{x}, \lambda). \qquad (9)$$

The proposed algorithm for stochastically optimizing the problem in (9) is summarized in Algorithm 1. It differs from the existing stochastic gradient descent methods in that it updates both the primal variable $\mathbf{x}$ (steps 4 and 5) and the dual variable $\lambda$ (step 6), which shares the same step sizes.

We note that the parameter $\rho$ is not employed in the implementation of Algorithm 1 and is only required for the theoretical analysis. It is noticeable that a similar primal-dual updating is explored in [15] to avoid projection in online learning. Our work differs from [15] in that their algorithm and analysis only lead to a bound for the regret and the violation of the constraints in a long run, which does not necessarily guarantee the feasibility of final solution. Also our proof techniques differ from [16], where the convergence rate is obtained for the saddle point; however our goal is to attain bound on the convergence of the primal feasible solution.

**Remark 2.** *The convex-concave optimization problem in (9) is equivalent to the following minimization problem:*

$$\min_{\mathbf{x} \in \mathcal{B}} \ f(\mathbf{x}) + \frac{[g(\mathbf{x})]_+^2}{2\gamma}, \tag{10}$$

*where $[z]_+$ outputs $z$ if $z > 0$ and zero otherwise. It thus may seem attractive to directly optimize the penalized function $f(\mathbf{x}) + [g(\mathbf{x})]_+^2/(2\gamma)$ using the standard SGD method, which unfortunately does not yield a regret of $O(\sqrt{T})$. This is because, in order to obtain a regret of $O(\sqrt{T})$, we need to set $\gamma = \Omega(\sqrt{T})$, which unfortunately will lead to a blowup of the gradients and consequently a poor regret bound. Using a primal-dual updating schema allows us to adjust the penalization term more carefully to obtain an $O(1/\sqrt{T})$ convergence rate.*

**Theorem 1.** *For any general convex function $f(\mathbf{x})$, if we set $\eta_t = \gamma/(2G_2^2), t = 1, \cdots, T$, and $\gamma = G_2^2/\sqrt{(G_1^2 + C_2^2 + (1 + \ln(2/\delta))\sigma^2)T}$ in Algorithm 1, under assumptions **A1-A3**, we have, with a probability at least $1 - \delta$,*

$$f(\widetilde{\mathbf{x}}_T) \leq \min_{\mathbf{x} \in \mathcal{K}} f(\mathbf{x}) + O\left(\frac{1}{\sqrt{T}}\right),$$

*where $O(\cdot)$ suppresses polynomial factors that depend on $\ln(2/\delta), G_1, G_2, C_2, \rho$, and $\sigma$.*

## 4.2 SGD with One Projection for Strongly Convex Optimization

We first emphasize that it is difficult to extend Algorithm 1 to achieve an $O(\ln T/T)$ convergence rate for strongly convex optimization. This is because although $-L(\mathbf{x}, \lambda)$ is strongly convex in $\lambda$, its modulus for strong convexity is $\gamma$, which is too small to obtain an $O(\ln T)$ regret bound.

To achieve a faster convergence rate for strongly convex optimization, we change assumptions **A1** and **A2** to

$$\mathbf{A4} \quad \|\widetilde{\nabla} f(\mathbf{x}, \xi_t)\|_2 \leq G_1, \quad \|\nabla g(\mathbf{x})\|_2 \leq G_2, \quad \forall \mathbf{x} \in \mathcal{B},$$

where we slightly abuse the same notation $G_1$. Note that **A1** only requires that $\|\nabla f(\mathbf{x})\|_2$ is bounded and **A2** assumes a mild condition on the stochastic gradient. In contrast, for strongly convex optimization we need to assume a bound on the stochastic gradient $\|\widetilde{\nabla} f(\mathbf{x}, \xi_t)\|_2$. Although assumption **A4** is stronger than assumptions **A1** and **A2**, however, it is always possible to bound the stochastic gradient for machine learning problems where $f(\mathbf{x})$ usually consists of a summation of loss functions on training examples, and the stochastic gradient is computed by sampling over the training examples. Given the bound on $\|\widetilde{\nabla} f(\mathbf{x}, \xi_t)\|_2$, we can easily have $\|\nabla f(\mathbf{x})\|_2 = \|\mathrm{E}\widetilde{\nabla} f(\mathbf{x}, \xi_t)\|_2 \leq \mathrm{E}\|\widetilde{\nabla} f(\mathbf{x}, \xi_t)\|_2 \leq G_1$, which is used to set an input parameter $\lambda_0 > G_1/\rho$ to the algorithm. According to the discussion in the last subsection, we know that the optimal dual variable $\lambda_*$ is upper bounded by $G_1/\rho$, and consequently is upper bounded by $\lambda_0$.

Similar to the last approach, we write the optimization problem (1) into an equivalent convex-concave optimization problem:

$$\min_{g(\mathbf{x}) \leq 0} f(\mathbf{x}) = \min_{\mathbf{x} \in \mathcal{B}} \max_{0 \leq \lambda \leq \lambda_0} f(\mathbf{x}) + \lambda g(\mathbf{x}) = \min_{\mathbf{x} \in \mathcal{B}} f(\mathbf{x}) + \lambda_0 [g(\mathbf{x})]_+.$$

To avoid unnecessary complication due to the subgradient of $[\cdot]_+$, following [18], we introduce a smoothing term $H(\lambda/\lambda_0)$, where $H(p) = -p \ln p - (1 - p) \ln(1 - p)$ is the entropy function, into the Lagrangian function, leading to the optimization problem $\min_{\mathbf{x} \in \mathcal{B}} F(\mathbf{x})$, where $F(\mathbf{x})$ is defined as

$$F(\mathbf{x}) = f(\mathbf{x}) + \max_{0 \leq \lambda \leq \lambda_0} \lambda g(\mathbf{x}) + \gamma H(\lambda/\lambda_0) = f(\mathbf{x}) + \gamma \ln\left(1 + \exp\left(\frac{\lambda_0 g(\mathbf{x})}{\gamma}\right)\right),$$

where $\gamma > 0$ is a parameter whose value will be determined later. Given the smoothed objective function $F(\mathbf{x})$, we find the optimal solution by applying SGD to minimize $F(\mathbf{x})$, where the gradient

---

**Algorithm 2** (SGDP-ST): SGD with ONE Projection by a Smoothing Technique

---
1: **Input**: a sequence of step sizes $\{\eta_t\}$, $\lambda_0$, and $\gamma$
2: **Initialization:** $\mathbf{x}_1 = \mathbf{0}$.
3: **for** $t = 1, \ldots, T$ **do**
4:   Compute $\mathbf{x}'_{t+1} = \mathbf{x}_t - \eta_t \left( \widetilde{\nabla} f(\mathbf{x}_t, \xi_t) + \dfrac{\exp\left(\lambda_0 g(\mathbf{x}_t)/\gamma\right)}{1 + \exp(\lambda_0 g(\mathbf{x}_t)/\gamma)} \lambda_0 \nabla g(\mathbf{x}_t) \right)$
5:   Update $\mathbf{x}_{t+1} = \mathbf{x}'_{t+1} / \max(\|\mathbf{x}'_{t+1}\|_2, 1)$
6: **end for**
7: **Output:** $\widetilde{\mathbf{x}}_T = \Pi_\mathcal{K}(\widehat{\mathbf{x}}_T)$, where $\widehat{\mathbf{x}}_T = \sum_{t=1}^{T} \mathbf{x}_t/T$.

---

of $F(\mathbf{x})$ is computed by

$$\nabla F(\mathbf{x}) = \nabla f(\mathbf{x}) + \frac{\exp\left(\lambda_0 g(\mathbf{x})/\gamma\right)}{1 + \exp\left(\lambda_0 g(\mathbf{x})/\gamma\right)} \lambda_0 \nabla g(\mathbf{x}). \tag{11}$$

Algorithm 2 gives the detailed steps. Unlike Algorithm 1, only the primal variable $\mathbf{x}$ is updated in each iteration using the stochastic gradient computed in (11).

The following theorem shows that Algorithm 2 achieves an $O(\ln T/T)$ convergence rate if the cost functions are strongly convex.

**Theorem 2.** *For any $\beta$-strongly convex function $f(\mathbf{x})$, if we set $\eta_t = 1/(2\beta t), t = 1, \ldots, T$, $\gamma = \ln T/T$, and $\lambda_0 > G_1/\rho$ in Algorithm 2, under assumptions **A3** and **A4**, we have with a probability at least $1 - \delta$,*

$$f(\widetilde{\mathbf{x}}_T) \le \min_{\mathbf{x} \in \mathcal{K}} f(\mathbf{x}) + O\left(\frac{\ln T}{T}\right),$$

*where $O(\cdot)$ suppresses polynomial factors that depend on $\ln(1/\delta)$, $1/\beta, G_1, G_2, \rho$, and $\lambda_0$.*

It is well known that the optimal convergence rate of SGD for strongly convex optimization is $O(1/T)$ [9] which has been proven to be tight in stochastic optimization setting [1]. According to Theorem 2, Algorithm 2 achieves an almost optimal convergence rate except for the factor of $\ln T$. It is worth mentioning that although it is not explicitly given in Theorem 2, the detailed expression for the convergence rate of Algorithm 2 exhibits a tradeoff in setting $\lambda_0$ (more can be found in the proof of Theorem 2). Finally, under assumptions **A1**-**A3**, Algorithm 2 can achieve an $O(1/\sqrt{T})$ convergence rate for general convex functions, similar to Algorithm 1.

## 5 Convergence Rate Analysis

We here present the proofs of main theorems. The omitted proofs are provided in the Appendix. We use $O(\cdot)$ notation in a few inequalities to absorb constants independent from $T$ for ease of exposition.

### 5.1 Proof of Theorem 1

To pave the path for the proof, we present a series of lemmas. The lemma below states two key inequalities, which follows the standard analysis of gradient descent.

**Lemma 1.** *Under the bounded assumptions in (6) and (7), for any $\mathbf{x} \in \mathcal{B}$ and $\lambda > 0$, we have*

$$(\mathbf{x}_t - \mathbf{x})^\top \nabla_\mathbf{x} L(\mathbf{x}_t, \lambda_t) \le \frac{1}{2\eta_t} \left( \|\mathbf{x} - \mathbf{x}_t\|_2^2 - \|\mathbf{x} - \mathbf{x}_{t+1}\|_2^2 \right) + 2\eta_t G_1^2 + \eta_t G_2^2 \lambda_t^2$$
$$+ 2\eta_t \underbrace{\|\widetilde{\nabla} f(\mathbf{x}_t, \xi_t) - \nabla f(\mathbf{x}_t)\|_2^2}_{\equiv \Delta_t} + \underbrace{(\mathbf{x} - \mathbf{x}_t)^\top (\widetilde{\nabla} f(\mathbf{x}_t, \xi_t) - \nabla f(\mathbf{x}_t))}_{\equiv \zeta_t(\mathbf{x})},$$

$$(\lambda - \lambda_t) \nabla_\lambda L(\mathbf{x}_t, \lambda_t) \le \frac{1}{2\eta_t} \left( |\lambda - \lambda_t|^2 - |\lambda - \lambda_{t+1}|^2 \right) + 2\eta_t C_2^2.$$

An immediate result of Lemma 1 is the following which states a regret-type bound.

**Lemma 2.** *For any general convex function $f(\mathbf{x})$, if we set $\eta_t = \gamma/(2G_2^2), t = 1, \cdots, T$, we have*

$$\sum_{t=1}^{T} (f(\mathbf{x}_t) - f(\mathbf{x}^*)) + \frac{[\sum_{t=1}^{T} g(\mathbf{x}_t)]_+^2}{2(\gamma T + 2G_2^2/\gamma)} \le \frac{G_2^2}{\gamma} + \frac{(G_1^2 + C_2^2)}{G_2^2} \gamma T + \frac{\gamma}{G_2^2} \sum_{t=1}^{T} \Delta_t + \sum_{t=1}^{T} \zeta_t(\mathbf{x}^*),$$

*where $\mathbf{x}^* = \arg\min_{\mathbf{x} \in \mathcal{K}} f(\mathbf{x})$.*

*Proof of Therorem 1.* First, by martingale inequality (e.g., Lemma 4 in [13]), with a probability $1 - \delta/2$, we have $\sum_{t=1}^{T} \zeta_t(\mathbf{x}^*) \leq 2\sigma\sqrt{3\ln(2/\delta)}\sqrt{T}$. By Markov's inequality, with a probability $1 - \delta/2$, we have $\sum_{t=1}^{T} \Delta_t \leq (1 + \ln(2/\delta))\sigma^2 T$. Substituting these inequalities into Lemma 2, plugging the stated value of $\gamma$, we have with a probability $1 - \delta$

$$\sum_{t=1}^{T}(f(\mathbf{x}_t) - f(\mathbf{x}^*)) + \frac{1}{C\sqrt{T}}\big[\sum_{t=1}^{T} g(\mathbf{x}_t)\big]_+^2 \leq O(\sqrt{T}),$$

where $C = 2G_2(1/\sqrt{G_1^2 + C_2^2 + (1 + \ln(2/\delta))\sigma^2} + 2\sqrt{G_1^2 + C_2^2 + (1 + \ln(2/\delta))\sigma^2})$ and $O(\cdot)$ suppresses polynomial factors that depend on $\ln(2/\delta), G_1, G_2, C_2, \sigma$.

Recalling the definition of $\widehat{\mathbf{x}}_T = \sum_{t=1}^{T} \mathbf{x}_t/T$ and using the convexity of $f(\mathbf{x})$ and $g(\mathbf{x})$, we have

$$f(\widehat{\mathbf{x}}_T) - f(\mathbf{x}^*) + \frac{\sqrt{T}}{C}[g(\widehat{\mathbf{x}}_T)]_+^2 \leq O\left(\frac{1}{\sqrt{T}}\right). \tag{12}$$

Assume $g(\widehat{\mathbf{x}}_T) > 0$, otherwise $\widetilde{\mathbf{x}}_T = \widehat{\mathbf{x}}_T$ and we easily have $f(\widetilde{\mathbf{x}}_T) \leq \min_{\mathbf{x}\in\mathcal{K}} f(\mathbf{x}) + O(1/\sqrt{T})$. Since $\widetilde{\mathbf{x}}_T$ is the projection of $\widehat{\mathbf{x}}_T$ into $\mathcal{K}$, i.e., $\widetilde{\mathbf{x}}_T = \arg\min_{g(\mathbf{x})\leq 0} \|\mathbf{x} - \widehat{\mathbf{x}}_T\|_2^2$, then by first order optimality condition, there exists a positive constant $s > 0$ such that

$$g(\widetilde{\mathbf{x}}_T) = 0, \text{ and } \widehat{\mathbf{x}}_T - \widetilde{\mathbf{x}}_T = s\nabla g(\widetilde{\mathbf{x}}_T)$$

which indicates that $\widehat{\mathbf{x}}_T - \widetilde{\mathbf{x}}_T$ is in the same direction to $\nabla g(\tilde{\mathbf{x}}_T)$. Hence,

$$g(\widehat{\mathbf{x}}_T) = g(\widehat{\mathbf{x}}_T) - g(\widetilde{\mathbf{x}}_T) \geq (\widehat{\mathbf{x}}_T - \widetilde{\mathbf{x}}_T)^\top \nabla g(\widetilde{\mathbf{x}}_T) = \|\widehat{\mathbf{x}}_T - \widetilde{\mathbf{x}}_T\|_2 \|\nabla g(\widetilde{\mathbf{x}}_T)\|_2 \geq \rho\|\widehat{\mathbf{x}}_T - \widetilde{\mathbf{x}}_T\|_2, \tag{13}$$

where the last inequality follows the definition of $\min_{g(\mathbf{x})=0} \|\nabla g(\mathbf{x})\|_2 \geq \rho$. Additionally, we have

$$f(\mathbf{x}^*) - f(\widehat{\mathbf{x}}_T) \leq f(\mathbf{x}^*) - f(\widetilde{\mathbf{x}}_T) + f(\widetilde{\mathbf{x}}_T) - f(\widehat{\mathbf{x}}_T) \leq G_1\|\widehat{\mathbf{x}}_T - \widetilde{\mathbf{x}}_T\|_2, \tag{14}$$

due to $f(\mathbf{x}^*) \leq f(\widetilde{\mathbf{x}}_T)$ and Lipschitz continuity of $f(\mathbf{x})$. Combining inequalities (12), (13), and (14) yields

$$\frac{\rho^2}{C}\sqrt{T}\|\widehat{\mathbf{x}}_T - \widetilde{\mathbf{x}}_T\|_2^2 \leq O(1/\sqrt{T}) + G_1\|\widehat{\mathbf{x}}_T - \widetilde{\mathbf{x}}_T\|_2.$$

By simple algebra, we have $\|\widehat{\mathbf{x}}_T - \widetilde{\mathbf{x}}_T\|_2 \leq \frac{G_1 C}{\rho^2\sqrt{T}} + O\left(\sqrt{\frac{C}{\rho^2 T}}\right)$. Therefore

$$f(\widetilde{\mathbf{x}}_T) \leq f(\widetilde{\mathbf{x}}_T) - f(\widehat{\mathbf{x}}_T) + f(\widehat{\mathbf{x}}_T) \leq G_1\|\widehat{\mathbf{x}}_T - \widetilde{\mathbf{x}}_T\|_2 + f(\mathbf{x}^*) + O\left(\frac{1}{\sqrt{T}}\right) \leq f(\mathbf{x}^*) + O\left(\frac{1}{\sqrt{T}}\right),$$

where we use the inequality in (12) to bound $f(\widehat{\mathbf{x}}_T)$ by $f(\mathbf{x}^*)$ and absorb the dependence on $\rho, G_1, C$ into the $O(\cdot)$ notation. $\square$

**Remark 3.** *From the proof of Theorem 1, we can see that the key inequalities are (12), (13), and (14). In particular, the regret-type bound in (12) depends on the algorithm. If we only update the primal variable using the penalized objective in (10), whose gradient depends on $1/\gamma$, it will cause a blowup in the regret bound with $(1/\gamma + \gamma T + T/\gamma)$, which leads to a non-convergent bound.*

### 5.2 Proof of Theorem 2

Our proof of Theorem 2 for the convergence rate of Algorithm 2 when applied to strongly convex functions starts with the following lemma by analogy of Lemma 2.

**Lemma 3.** *For any $\beta$-strongly convex function $f(\mathbf{x})$, if we set $\eta_t = 1/(2\beta t)$, we have*

$$\sum_{t=1}^{T}(F(\mathbf{x}) - F(\mathbf{x}^*)) \leq \frac{(G_1^2 + \lambda_0^2 G_2^2)(1 + \ln T)}{2\beta} + \sum_{t=1}^{T}\zeta_t(\mathbf{x}^*) - \frac{\beta}{4}\sum_{t=1}^{T}\|\mathbf{x}^* - \mathbf{x}_t\|_2^2$$

*where $\mathbf{x}^* = \arg\min_{\mathbf{x}\in\mathcal{K}} f(\mathbf{x})$.*

In order to prove Theorem 2, we need the following result for an improved martingale inequality.

**Lemma 4.** *For any fixed $\mathbf{x} \in \mathcal{B}$, define $D_T = \sum_{t=1}^{T}\|\mathbf{x}_t - \mathbf{x}\|_2^2$, $\Lambda_T = \sum_{t=1}^{T}\zeta_t(\mathbf{x})$, and $m = \lceil\log_2 T\rceil$. We have*

$$\Pr\left(D_T \leq \frac{4}{T}\right) + \Pr\left(\Lambda_T \leq 4G_1\sqrt{D_T \ln\frac{m}{\delta}} + 4G_1\ln\frac{m}{\delta}\right) \geq 1 - \delta.$$

*Proof of Theorem 2.* We substitute the bound in Lemma 4 into the inequality in Lemma 3 with $\mathbf{x} = \mathbf{x}^*$. We consider two cases. In the first case, we assume $D_T \leq 4/T$. As a result, we have

$$\sum_{t=1}^{T} \zeta_t(\mathbf{x}^*) = \sum_{t=1}^{T} (\nabla f(\mathbf{x}_t) - \widetilde{\nabla} f(\mathbf{x}_t, \xi_t))^\top (\mathbf{x}^* - \mathbf{x}_t) \leq 2G_1 \sqrt{TD_T} \leq 4G_1,$$

which together with the inequality in Lemma 3 leads to the bound

$$\sum_{t=1}^{T} (F(\mathbf{x}_t) - F(\mathbf{x}^*)) \leq 4G_1 + \frac{(G_1^2 + \lambda_0^2 G_2^2)(1 + \ln T)}{2\beta}.$$

In the second case, we assume

$$\sum_{t=1}^{T} \zeta_t(\mathbf{x}^*) \leq 4G_1 \sqrt{D_T \ln \frac{m}{\delta}} + 4G_1 \ln \frac{m}{\delta} \leq \frac{\beta}{4} D_T + \left( \frac{16G_1^2}{\beta} + 4G_1 \right) \ln \frac{m}{\delta},$$

where the last step uses the fact $2\sqrt{ab} \leq a^2 + b^2$. We thus have

$$\sum_{t=1}^{T} (F(\mathbf{x}_t) - F(\mathbf{x}^*)) \leq \left( \frac{16G_1^2}{\beta} + 4G_1 \right) \ln \frac{m}{\delta} + \frac{(G_1^2 + \lambda_0^2 G_2^2)(1 + \ln T)}{2\beta}.$$

Combing the results of the two cases, we have, with a probability $1 - \delta$,

$$\sum_{t=1}^{T} (F(\mathbf{x}_t) - F(\mathbf{x}^*)) \leq \underbrace{\left( \frac{16G_1^2}{\beta} + 4G_1 \right) \ln \frac{m}{\delta} + 4G_1 + \frac{(G_1^2 + \lambda_0^2 G_2^2)(1 + \ln T)}{2\beta}}_{O(\ln T)}.$$

By convexity of $F(\mathbf{x})$, we have $F(\widehat{\mathbf{x}}_T) \leq F(\mathbf{x}^*) + O(\ln T/T)$. Noting that $\mathbf{x}^* \in \mathcal{K}$, $g(\mathbf{x}^*) \leq 0$, we have $F(\mathbf{x}^*) \leq f(\mathbf{x}^*) + \gamma \ln 2$. On the other hand,

$$F(\widehat{\mathbf{x}}_T) = f(\widehat{\mathbf{x}}_T) + \gamma \ln \left( 1 + \exp \left( \frac{\lambda_0 g(\widehat{\mathbf{x}}_T)}{\gamma} \right) \right) \geq f(\widehat{\mathbf{x}}_T) + \max (0, \lambda_0 g(\widehat{\mathbf{x}}_T)).$$

Therefore, with the value of $\gamma = \ln T/T$, we have

$$f(\widehat{\mathbf{x}}_T) \leq f(\mathbf{x}^*) + O\left( \frac{\ln T}{T} \right), \tag{15}$$

$$f(\widehat{\mathbf{x}}_T) + \lambda_0 g(\widehat{\mathbf{x}}_T) \leq f(\mathbf{x}^*) + O\left( \frac{\ln T}{T} \right). \tag{16}$$

Applying the inequalities (13) and (14) to (16), and noting that $\gamma = \ln T/T$, we have

$$\lambda_0 \rho \|\widehat{\mathbf{x}}_T - \widetilde{\mathbf{x}}_T\|_2 \leq G_1 \|\widehat{\mathbf{x}}_T - \widetilde{\mathbf{x}}_T\|_2 + O\left( \frac{\ln T}{T} \right).$$

For $\lambda_0 > G_1/\rho$, we have $\|\widehat{\mathbf{x}}_T - \widetilde{\mathbf{x}}_T\|_2 \leq (1/(\lambda_0 \rho - G_1))O(\ln T/T)$. Therefore

$$f(\widetilde{\mathbf{x}}_T) \leq f(\widetilde{\mathbf{x}}_T) - f(\widehat{\mathbf{x}}_T) + f(\widehat{\mathbf{x}}_T) \leq G_1 \|\widehat{\mathbf{x}}_T - \widetilde{\mathbf{x}}_T\|_2 + f(\mathbf{x}^*) + O\left( \frac{\ln T}{T} \right) \leq f(\mathbf{x}^*) + O\left( \frac{\ln T}{T} \right),$$

where in the second inequality we use inequality (15). $\qquad \square$

# 6 Conclusions

In the present paper, we made a progress towards making the SGD method efficient by proposing a framework in which it is possible to exclude the projection steps from the SGD algorithm. We have proposed two novel algorithms to overcome the computational bottleneck of the projection step in applying SGD to optimization problems with complex domains. We showed using novel theoretical analysis that the proposed algorithms can achieve an $O(1/\sqrt{T})$ convergence rate for general convex functions and an $O(\ln T/T)$ rate for strongly convex functions with a overwhelming probability which are known to be optimal (up to a logarithmic factor) for stochastic optimization.

### Acknowledgments

The authors would like to thank the anonymous reviewers for their helpful suggestions. This work was supported in part by National Science Foundation (IIS-0643494) and Office of Navy Research (Award N000141210431 and Award N00014-09-1-0663).

# References

[1] A. Agarwal, P. L. Bartlett, P. D. Ravikumar, and M. J. Wainwright. Information-theoretic lower bounds on the oracle complexity of stochastic convex optimization. *IEEE Transactions on Information Theory*, 58(5):3235–3249, 2012.

[2] F. Bach and E. Moulines. Non-asymptotic analysis of stochastic approximation algorithms for machine learning. In *NIPS*, pages 451–459, 2011.

[3] D. P. Bertsekas. *Nonlinear Programming*. Athena Scientific, 2nd edition, 1999.

[4] S. Boyd and L. Vandenberghe. *Convex Optimization*. Cambridge University Press, 2004.

[5] K. L. Clarkson. Coresets, sparse greedy approximation, and the frank-wolfe algorithm. *ACM Transactions on Algorithms*, 6(4), 2010.

[6] J. Duchi, S. Shalev-Shwartz, Y. Singer, and T. Chandra. Efficient projections onto the l1-ball for learning in high dimensions. In *ICML*, pages 272–279, 2008.

[7] M. Frank and P. Wolfe. An algorithm for quadratic programming. *Naval Research Logistics*, 3, 1956.

[8] E. Hazan. Sparse approximate solutions to semidefinite programs. In *LATIN*, pages 306–316, 2008.

[9] E. Hazan and S. Kale. Beyond the regret minimization barrier: an optimal algorithm for stochastic strongly-convex optimization. *Journal of Machine Learning Research - Proceedings Track*, 19:421–436, 2011.

[10] E. Hazan and S. Kale. Projection-free online learning. In *ICML*, 2012.

[11] M. Jaggi. *Sparse Convex Optimization Methods for Machine Learning*. PhD thesis, ETH Zurich, Oct. 2011.

[12] M. Jaggi and M. Sulovský. A simple algorithm for nuclear norm regularized problems. In *ICML*, pages 471–478, 2010.

[13] G. Lan. An optimal method for stochastic composite optimization. *Math. Program.*, 133(1-2):365–397, 2012.

[14] J. Liu and J. Ye. Efficient euclidean projections in linear time. In *ICML*, page 83, 2009.

[15] M. Mahdavi, R. Jin, and T. Yang. Trading regret for efficiency: online convex optimization with long term constraints. *JMLR*, 13:2465–2490, 2012.

[16] A. Nemirovski, A. Juditsky, G. Lan, and A. Shapiro. Robust stochastic approximation approach to stochastic programming. *SIAM J. on Optimization*, 19:1574–1609, 2009.

[17] Y. Nesterov. *Introductory Lectures on Convex Optimization: A Basic Course*. Kluwer Academic Publishers, 2004.

[18] Y. Nesterov. Smooth minimization of non-smooth functions. *Math. Program.*, 103(1):127–152, 2005.

[19] S. Shalev-Shwartz, Y. Singer, and N. Srebro. Pegasos: Primal estimated sub-gradient solver for svm. In *ICML*, pages 807–814, 2007.

[20] Y. Ying and P. Li. Distance metric learning with eigenvalue optimization. *JMLR.*, 13:1–26, 2012.

[21] T. Zhang. Sequential greedy approximation for certain convex optimization problems. *Information Theory, IEEE Transactions on*, 49:682–691, 2003.

[22] M. Zinkevich. Online convex programming and generalized infinitesimal gradient ascent. In *ICML*, pages 928–936, 2003.

